# Edge of Chaos Computation in Mixed-Mode VLSI - "A Hard Liquid"

**Felix Schürmann, Karlheinz Meier, Johannes Schemmel**
Kirchhoff Institute for Physics
University of Heidelberg
Im Neuenheimer Feld 227, 69120 Heidelberg, Germany
`felix.schuermann@kip.uni-heidelberg.de`,
WWW home page: `http://www.kip.uni-heidelberg.de/vision`

## Abstract

Computation without stable states is a computing paradigm different from Turing's and has been demonstrated for various types of simulated neural networks. This publication transfers this to a hardware implemented neural network. Results of a software implementation are reproduced showing that the performance peaks when the network exhibits dynamics at the edge of chaos. The *liquid computing* approach seems well suited for operating analog computing devices such as the used VLSI neural network.

## 1   Introduction

Using artificial neural networks for problem solving immediately raises the issue of their general trainability and the appropriate learning strategy. Topology seems to be a key element, especially, since algorithms do not necessarily perform better when the size of the network is simply increased. Hardware implemented neural networks, on the other hand, offer scalability in complexity and gain in speed but naturally do not compete in flexibility with software solutions. Except for specific applications or highly iterative algorithms [1], the capabilities of hardware neural networks as generic problem solvers are difficult to assess in a straight-forward fashion.

Independently, Maass et al.[2] and Jaeger [3] proposed the idea of computing without stable states. They both used randomly connected neural networks as non-linear dynamical systems with the inputs causing perturbations to the transient response of the network. In order to customize such a system for a problem, a readout is trained which requires only the network reponse of a single time step for input. The readout may be as simple as a linear classifier: 'training' then reduces to a well defined least-squares linear regression. Justification for this splitting into a non-linear transformation followed by a linear one originates from Cover [4]. He proved that the probability for a pattern classification problem to be linearly separable is higher when cast in a high-dimensional space by a non-linear mapping.

In the terminology of Maass et al., the non-linear dynamical system is called a *liquid* and together with the readouts it represents a *liquid state machine* (LSM).

It has been proven that under certain conditions the LSM concept is universal on functions of time [2].

Adopting the liquid computing strategy for mixed-mode hardware implemented networks using very large scale integration (VLSI) offers two promising prospects: First, such a system profits immediately from scaling, i.e., more neurons increase the complexity of the network dynamics while not increasing training complexity. Second, it is expected that the liquid approach can cope with an imperfect substrate as commonly present in analog hardware. Configuring highly integrated analog hardware as a liquid therefore seems a promising way for analog computing. This conclusion is not unexpected since the liquid computing paradigm was inspired by a complex and 'analog' system in the first place: the biological nervous system [2].

This publication presents initial results on configuring a general purpose mixed-mode neural network ASIC (application specific integrated circuit) as a liquid. The used custom-made ANN ASIC [5] provides 256 McCulloch-Pitts neurons with about 33k analog synapses and allows a wide variety of topologies, especially highly recurrent ones. In order to operate the ASIC as a liquid a generation procedure proposed by Bertschinger et al. [6] is adopted that generates the network topology and weights. These authors as well showed that the performance of those input-driven networks—meant are the suitable properties of the network dynamics to act as a liquid—depends on whether the response of the liquid to the inputs is ordered or chaotic. Precisely, according to a special measure the performance peaks when the liquid is *inbetween* order and chaos. The reconfigurability of the used ANN ASIC allows to explore various generation parameters, i.e., physically different liquids are evaluated; the obtained experimental results are in accordance with the previously published software simulations [6].

## 2 Substrate

The substrate used in the following is a general purpose ANN ASIC manufactured in a $0.35\mu$m CMOS process [5]. Its design goals were to implement small synapses while being fast reconfigurable and capable of operating at high speed; it therefore combines analog computation with digital signaling. It is comprised of 33k analog synapses with capacitive weight storage (nominal 10-bit plus sign) and 256 McCulloch-Pitts neurons. For efficiency it employs mostly current mode circuits. Experimental benchmark results using evolutionary algorithms training strategies have previously been published [1]. A full weight refresh can be performed within $200\mu$s and in the current setup one network cycle, i.e., the time base of the liquid, lasts about $0.5\mu$s. This is due to the prototype nature of the ASIC and its input/output; the core can already be operated about 20 times faster.

The analog operation of the chip is limited to the synaptic weights $\omega_{ij}$ and the input stage of the output neurons. Since both, input ($I_j$) and output signals ($O_i$) of the network are binary, the weight multiplication is reduced to a summation and the activation function $g(x)$ of the output neurons equals the Heaviside function $\Theta(x)$:

$$O_i = g(\sum_j \omega_{ij} I_j), \quad g(x) = \Theta(x),\ I, O \in \{0, 1\}. \tag{1}$$

The neural network chip is organized in four identical blocks; each represents a fully connected one-layer perceptron with McCulloch-Pitts neurons. One block basically consists of 128×64 analog synapses that connect each of the 128 inputs to each of the 64 output neurons. The network operates in a discrete time update scheme, i.e., Eq. 1 is calculated once for each network cycle. By feeding outputs back to the

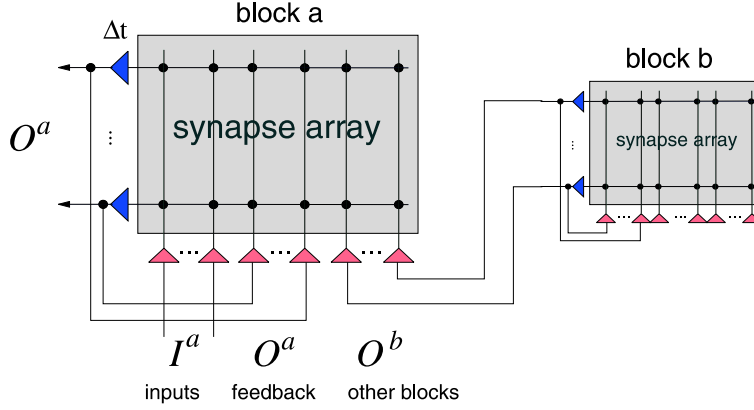

Figure 1: Network blocks can be configured for different input sources.

inputs a block can be configured as a recurrent network (c.f. Fig. 1). Additionally, outputs of the other network blocks can be fed back to the block's input. In this case the output of a neuron at time $t$ depends not only on the actual input but also on the previous network cycle and the activity of the other blocks. Denoting the time needed for one network cycle with $\Delta t$, the output function of one network block becomes:

$$O(t + \Delta t)_i^a = \Theta \left( \sum_j \omega_{ij} I(t)_j^a + \sum_{x \in \{a,b,c,d\}} \sum_k \omega_{ik}^x O(t)_k^x \right). \qquad (2)$$

Here, $\Delta t$ denotes the time needed for one network cycle. The first term in the argument of the activation function is the external input to the network block $I_j^a$. The second term models the feedback path from the output of block $a$, $O_k^a$, as well as the other 3 blocks $b,c,d$ back to its input. For two network blocks this is illustrated in Fig. 1. Principally, this model allows an arbitrarily large network that operates synchronously at a common network frequency $f_{net} = 1/\Delta t$ since the external input can be the output of other identical network chips.

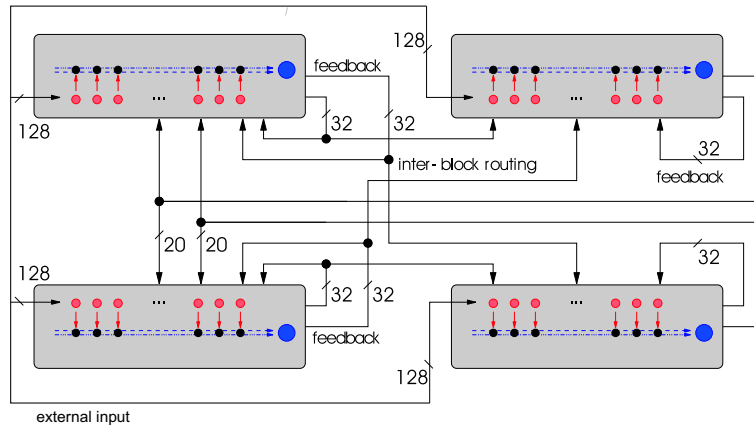

Figure 2: Intra- and inter-block routing schematic of the used ANN ASIC.

For the following experiments one complete ANN ASIC is used. Since one output

neuron has 128 inputs, it cannot be connected to all 256 neurons simultaneously. Furthermore, it can only make arbitrary connections to neurons of the same block, whereas the inter-block feedback fixes certain output neurons to certain inputs. Details of the routing are illustrated in Fig. 2.

The ANN ASIC is connected to a standard PC with a custom-made PCI-based interface card using a programmable logic to control the neural network chip.

## 3  Liquid Computing Setup

Following the terminology introduced by Maass et al. the ANN ASIC represents the liquid. Appropriately configured, it acts as a non-linear filter to the input. The response of the neural network ASIC at a certain time step is called the *liquid state* $x(t)$. This output is provided to the readout. In our case these are one or more linear classifiers implemented in software. The classifier result, and thus the response of the liquid state machine at a time $t$, is given by:

$$v(t) = \Theta(\sum w_i x_i(t)). \tag{3}$$

The weights $w_i$ are determined with a least-squares linear regression calculated for the desired target values $y(t)$. Using the same liquid state $x(t)$ multiple readouts can be used to predict differing target functions simultaneously (c.f. Fig. 3).

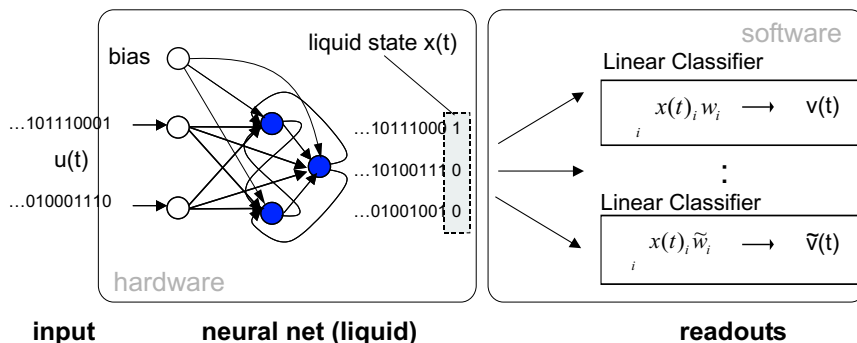

Figure 3: The liquid state machine setup.

The used setup is similar to the one used by Bertschinger et al. [6] with the central difference that the liquid here is implemented in hardware. The specific hardware design imposes McCulloch-Pitts type neurons that are either on or off ($O \in \{0, 1\}$) and not symmetric ($O \in \{-1, 1\}$). Besides of this, the topology and weight configuration of the ANN ASIC follow the procedure used by Bertschinger et al. The random generation of such input-driven networks is governed by the following parameters: $N$, the number of neurons; $k$, the number of incoming connections per neuron; $\sigma^2$, the variance of the zero-centered Gaussian distribution from which the weights for the incoming connections are drawn; $u(t)$, the external input signal driving each neuron. Bertschinger et al. used a random binary input signal $u(t)$ which assumes with equal chance $\overline{u} + 1$ or $\overline{u} - 1$. Since the used ANN ASIC has a fixed dynamic range for a single synapse, a weight can assume a normalized value in the interval $[-1, 1]$ with 11 bit accuracy. For this reason, the input signal $u(t)$ is split to a constant bias part $\overline{u}$ and the varying part, which again is split to an excitatory and its inverse contribution. Each neuron of the network then gets $k$ inputs from other neurons, one constant bias of weight $\overline{u}$, and two mutually exclusive input neurons

with weights 0.5 and $-0.5$. The latter modification was introduced to account for the fact that the inner neurons assume only the values $\{0, 1\}$. Using the input and its inverse accordingly recovers a differential weight change of 1 between the active and inactive state.

The performance of the liquid state machine is evaluated according to the mutual information of the target values $y(t)$ and the predicted values $v(t)$. This measure is defined as:

$$MI(v, y) = \sum_{v'} \sum_{y'} p(v', y') \log_2 \frac{p(v', y')}{p(v')p(y')}, \tag{4}$$

where $p(v') = \text{probability}\{v(t) = v'\}$ with $v' \in \{0, 1\}$ and $p(v', y')$ is the joint probability. It can be calculated from the confusion matrix of the linear classifier and can be given the dimension *bits*.

In order to assess the capability to account for inputs of preceeding time steps, it is sensible to define another measure, the memory capacity MC (cf. [7]):

$$MC = \sum_{\tau=0}^{\infty} MI(v_\tau, y_\tau). \tag{5}$$

Here, $v_\tau$ and $y_\tau$ denote the prediction and target shifted in time by $\tau$ time steps (i.e. $y_\tau(t) = y(t - \tau)$). It is as well measured in bits.

## 4    Results

A linear classifier by definition cannot solve a linearily non-separable problem. It therefore is a good test for the non-trivial contribution of the liquid if a liquid state machine with a linear readout has to solve a linearly non-separable problem. The benchmark problem used in the following is 3-bit parity in time, i.e., $y_\tau(t) = PARITY(u(t - \tau), u(t - \tau - 1), u(t - \tau - 2))$, which is known to be linearly non-separable. The linear classifiers are trained to predict the linearly non-separable $y_\tau(t)$ simply from the liquid state $x(t)$. To do this it is necessary that in the liquid state at time $t$ there is information present of the previous time steps.

Bertschinger et al. showed theoretically and in simulation that depending on the parameters $k$, $\sigma^2$, and $\overline{u}$ an input-driven neural network shows ordered or chaotic dynamics. This causes input information either to disappear quickly (the simplest case would be an identity map from input to output) or stay forever in the network respectively. Although the transition of the network dynamics from order to chaos happens gradually with the variation of the generation parameters $(k, \sigma^2, \overline{u})$, the performance as a liquid shows a distinct peak when the network exhibits dynamics inbetween order and chaos. These critical dynamics suggest the term "computation at the edge of chaos" which is originated by Langton [8].

The following results are obtained using the ANN ASIC as the liquid on a random binary input string $(u(t))$ of length 4000 for which the linear classifier is calculated. The shown mutual information and memory capacity are the measured performance on a random binary test string of length 8000. For each time shift $\tau$, a separate classifier is calculated. For each parameter set $k$, $\sigma^2$, $\overline{u}$ this procedure is repeated several times (for exact numbers compare the individual plots), i.e. several liquids are generated.

Fig. 4 shows the mutual information MI versus the shift in time $\tau$ for the 3-bit delayed parity problem and the network parameters fixed to $N = 256$, $k = 6$, $\sigma^2 = 0.14$, and $\overline{u} = 0$. Plotted are the mean values of 50 liquids evaluated in

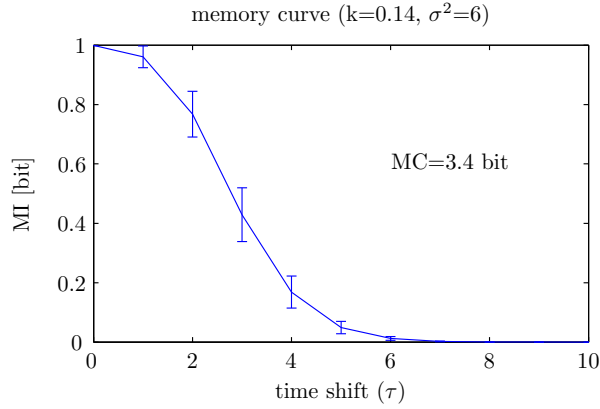

Figure 4: The mutual information between prediction and target for the 3-bit de-layed parity problem versus the delay for k=6, $\sigma^2$=0.14). The plotted limits are the 1-sigma spreads of 50 different liquids. The integral under this curve is the mean MC and is the maximum in the left plot of Fig. 5.

hardware and the given limits are the standard deviation in the mean. From the error limits it can be inferred that the parity problem is solved in all runs for $\tau = 0$, and in some for $\tau = 1$. For larger time shifts the performance decreases until the liquid has no information on the input anymore.

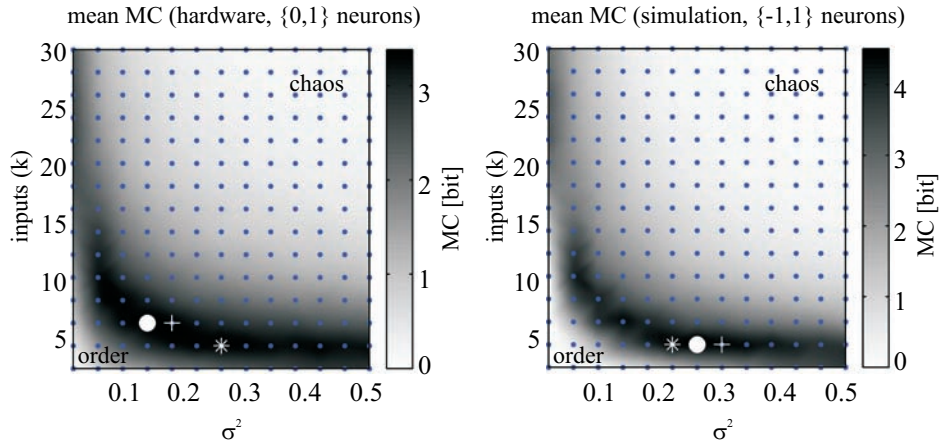

Figure 5: Shown are two parameter sweeps for the 3-bit delayed parity in depen-dence of the generation parameters $k$ and $\sigma^2$ with fixed $N = 256$, $\overline{u} = 0$. **Left:** 50 liquids per parameter set evaluated in hardware. **Right:** 35 liquids per parameter set using software simulation of ASIC but with symmetric neurons. Actual data points are marked with black dots, the gray shading shows an interpolation. The largest three mean MCs are marked with a white dot, asterisk, plus sign.

In order to assess how different generation parameters influence the quality of the liquid, parameter sweeps are performed. For each parameter set several liquids are generated and readouts trained. The obtained memory capacities of the runs are averaged and used as the performance measure. Fig. 5 shows a parameter sweep of $k$ and $\sigma^2$ for the memory capacity MC for $N = 256$, and $\overline{u} = 0$. On the left side, results obtained with the hardware are shown. The shading shows an interpolation

of the actual measured values marked with dots. The largest three mean MCs are marked in order with a white circle, white asterisk, and white plus.

It can be seen that the memory capacity peaks distinctly along a hyperbola-like band. The area below the transition band goes along with ordered dynamics; above it, the network exhibits chaotic behavior. The shape of the transition indicates a constant network activity for critical dynamics. The standard deviation in the mean of 50 liquids per parameter set is below 2%, i.e., the transition is significant.

The transition is not shown in a $\overline{u}$-$\sigma^2$-sweep as originally by Bertschinger et al. because in the hardware setup only a limited parameter range of $\sigma^2$ and $\overline{u}$ is accessible due to synapses of the range $[-1, 1]$ with a limited resolution. The accessible region ($\sigma^2 \in [0, 1]$ and $\overline{u} \in [0, 1]$) nonetheless exhibits a similar transition as described by Bertschinger et al. (not shown).

The smaller overall performance in memory capacity compared to their liquids, on the other hand, is simply due to the anti-symmetric neurons and not to other hardware restrictions as it can be seen from the right side of Fig. 5. There the same parameter sweep is shown, but this time the liquid is implemented in a software simulation of the ASIC with *symmetric* neurons. While all connectivity constraints of the hardware are incorporated in the simulation, the only other change in the setup is the adjustment of the input signal to $\overline{u} \pm 1$. 35 liquids per parameter set are evaluated. The observed performance decrease results from the asymmetry of the 0,1 neurons; a similar effect is observed by Bertschinger et al. for $\overline{u} \neq 0$.

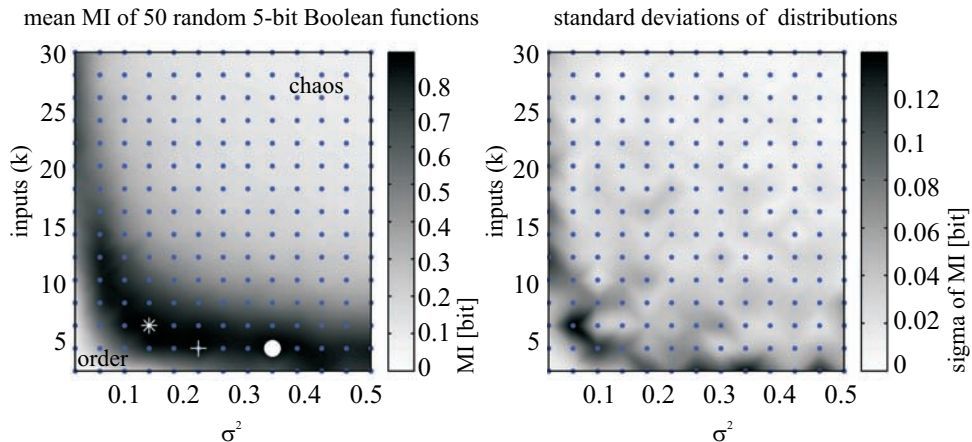

Figure 6: Mean mutual information of 50 simultaneously trained linear classifiers on randomly drawn 5-bit Boolean functions using the hardware liquid (10 liquids per parameter set evaluated). The right plot shows the 1-sigma spreads.

Finally, the hardware-based liquid state machine was tested on 50 randomly drawn Boolean functions of the last 5 inputs (5 bit in time) (cf. Fig. 6). In this setup, 50 linear classifiers read out the same liquid simultaneously to calculate their independent predictions at each time step. The mean mutual information ($\tau = 0$) for the 50 classifiers in 10 runs is plotted. From the right plot it can be seen that the standard deviation for the single measurement along the critical line is fairly small; this shows that critical dynamics yield a generic liquid independent of the readout.

# 5 Conclusions & Outlook

Computing without stable states manifests a new computing paradigm different to the Turing approach. By different authors this has been investigated for various types of neural networks, theoretically and in software simulation. In the present publication these ideas are transferred back to an analog computing device: a mixed-mode VLSI neural network. Earlier published results of Bertschinger et al. were reproduced showing that the readout with linear classifiers is especially successful when the network exhibits critical dynamics.

Beyond the point of solving rather academic problems like 3-bit parity, the liquid computing approach may be well suited to make use of the massive resources found in analog computing devices, especially, since the liquid is generic, i.e. independent of the readout. The experiments with the general purpose ANN ASIC allow to explore the necessary connectivity and accuracy of future hardware implementations. With even higher integration densities the inherent unreliability of the elementary parts of VLSI systems grows, making fault-tolerant training and operation methods necessary. Even though it has not be shown in this publication, initial experiments support that the used liquids show a robustness against faults introduced after the readout has been trained.

As a next step it is planned to use parts of the ASIC to realize the readout. Such a liquid state machine can make use of the hardware implementation and will be able to operate in real-time on continuous data streams.

## References

[1] S. Hohmann, K. Fieres, J. Meier, T. Schemmel, J. Schmitz, and F. Schürmann. Training fast mixed-signal neural networks for data classification. In *Proceedings of the International Joint Conference on Neural Networks IJCNN'04*, pages 2647–2652. IEEE Press, July 2004.

[2] W. Maass, T. Natschläger, and H. Markram. Real-time computing without stable states: A new framework for neural computation based on perturbations. *Neural Computation*, 14(11):2531–2560, 2002.

[3] H. Jaeger. The "echo state" approach to analysing and training recurrent neural networks. Technical Report GMD Report 148, German National Research Center for Information Technology, 2001.

[4] T. M. Cover. Geometrical and statistical properties of systems of linear inequalities with application in pattern recognition. *IEEE Transactions on Electronic Computers*, EC-14:326–334, 1965.

[5] J. Schemmel, S. Hohmann, K. Meier, and F. Schürmann. A mixed-mode analog neural network using current-steering synapses. *Analog Integrated Circuits and Signal Processing*, 38(2-3):233–244, February-March 2004.

[6] N. Bertschinger and T. Natschläger. Real-time computation at the edge of chaos in recurrent neural networks. *Neural Computation*, 16(7):1413 – 1436, July 2004.

[7] T. Natschläger and W. Maass. Information dynamics and emergent computation in recurrent circuits of spiking neurons. In Sebastian Thrun, Lawrence Saul, and Bernhard Schölkopf, editors, *Proc. of NIPS 2003, Advances in Neural Information Processing Systems 16*. MIT Press, Cambridge, MA, 2004.

[8] C. G. Langton. Computation at the edge of chaos. *Physica D*, 42, 1990.
